# Nonparanormal Belief Propagation (NPNBP)

**Gal Elidan**
Department of Statistics
Hebrew University
galel@huji.ac.il

**Cobi Cario**
School of Computer Science and Engineering
Hebrew University
cobi.cario@mail.huji.ac.il

## Abstract

The empirical success of the belief propagation approximate inference algorithm has inspired numerous theoretical and algorithmic advances. Yet, for continuous non-Gaussian domains performing belief propagation remains a challenging task: recent innovations such as nonparametric or kernel belief propagation, while useful, come with a substantial computational cost and offer little theoretical guarantees, even for tree structured models. In this work we present Nonparanormal BP for performing efficient inference on distributions parameterized by a Gaussian copulas network and *any* univariate marginals. For tree structured networks, our approach is guaranteed to be exact for this powerful class of non-Gaussian models. Importantly, the method is as efficient as standard Gaussian BP, and its convergence properties do not depend on the complexity of the univariate marginals, even when a nonparametric representation is used.

## 1 Introduction

Probabilistic graphical models [Pearl, 1988] are widely use to model and reason about phenomena in a variety of domains such as medical diagnosis, communication, machine vision and bioinformatics. The usefulness of such models in complex domains, where exact computations are infeasible, relies on our ability to perform efficient and reasonably accurate inference of marginal and conditional probabilities. Perhaps the most popular approximate inference algoritm for graphical models is belief propagation (BP) [Pearl, 1988]. Guaranteed to be exact for trees, it is the surprising performance of the method when applied to general graphs (e.g., [McEliece et al., 1998, Murphy and Weiss, 1999]) that has inspired numerous works ranging from attempts to shed theoretical light on propagation-based algorithms (e.g., [Weiss and Freeman, 2001, Heskes, 2004, Mooij and Kappen, 2005]) to a wide range of algorithmic variants and generalizations (e.g., [Yedidia et al., 2001, Wiegerinck and Heskes, 2003, Globerson and Jaakkola, 2007]).

In most works, the variables are either discrete or the distribution is assumed to be Gaussian [Weiss and Freeman, 2001]. However, many continuous real-world phenomenon are far from Gaussian, and can have a complex multi-modal structure. This has inspired several innovative and practically useful methods specifically aimed at the continuous non-Gaussian case such as expectation propagation [Minka, 2001], particle BP [Ihler and McAllester, 2009], nonparametric BP [Sudderth et al., 2010b], and kernel BP [Song et al., 2011]. Since these works are aimed at general unconstrained distributions, they all come at a substantial computational price. Further, little can be said a-priori about their expected performance even in tree structured models. Naturally, we would like an inference algorithm that is as general as possible while being as computationally convenient as simple Gaussian BP [Weiss and Freeman, 2001].

In this work we present Nonparanormal BP (NPNBP), an inference method that strikes a balance between these competing desiderata. In terms of generality, we focus on the flexible class of Copula Bayesian Networks (CBNs) [Elidan, 2010] that are defined via local Gaussian copula functions and *any* univariate densities (possible nonparametric). Utilizing the power of the copula framework [Nelsen, 2007], these models can capture complex multi-modal and heavy-tailed phenomena.

Figure 1: Samples from the bivariate Gaussian copula with correlation $\theta = 0.25$.
(left) with unit variance Gaussian and Gamma marginals; (right) with a mixture of Gaussian and exponential marginals.

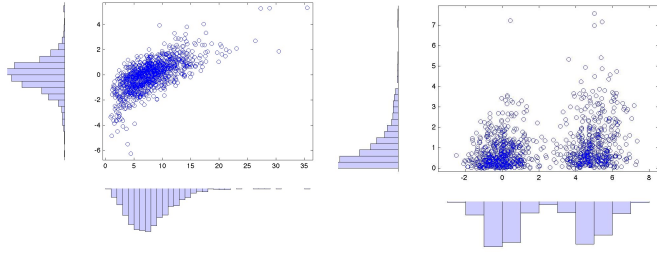

Algorithmically, our approach enjoys the benefits of Gaussian BP (GaBP). First, it is guaranteed to converge and return exact results on tree structured models, *regardless* of the form of the univariate densities. Second, it is computationally comparable to performing GaBP on a graph with the same structure. Third, its convergence properties on general graphs are similar to that of GaBP and, quite remarkably, *do not* depend on the complexity of the univariate marginals.

## 2  Background

In this section we provide a brief background on copulas in general, the Gaussian copula in particular, and the Copula Bayesian Network model of Elidan [2010].

### 2.1  The Gaussian Copula

A copula function [Sklar, 1959] links marginal distributions to form a multivariate one. Formally:

**Definition 2.1:** Let $U_1, \ldots, U_n$ be real random variables marginally uniformly distributed on $[0, 1]$. A copula function $C : [0, 1]^n \rightarrow [0, 1]$ is a joint distribution

$$C_\theta(u_1, \ldots, u_n) = P(U_1 \le u_1, \ldots, U_n \le u_n),$$

where $\theta$ are the parameters of the copula function. ∎

Now consider an arbitrary set $\mathcal{X} = \{X_1, \ldots X_n\}$ of real-valued random variables (typically *not* marginally uniformly distributed). Sklar's seminal theorem states that for *any* joint distribution $F_{\mathcal{X}}(\mathbf{x})$, there exists a copula function $C$ such that $F_{\mathcal{X}}(\mathbf{x}) = C(F_{X_1}(x_1), \ldots, F_{X_n}(x_n))$. When the univariate marginals are continuous, $C$ is uniquely defined.

The constructive converse, which is of central interest from a modeling perspective, is also true. Since $U_i \equiv F_i$ is itself a random variable that is always uniformly distributed in $[0, 1]$, *any* copula function taking *any* marginal distributions $\{U_i\}$ as its arguments, defines a valid joint distribution with marginals $\{U_i\}$. Thus, copulas are "distribution generating" functions that allow us to separate the choice of the univariate marginals and that of the dependence structure, encoded in the copula function $C$. Importantly, this flexibility often results in a construction that is beneficial in practice.

**Definition 2.2:** The Gaussian copula distribution is defined by:

$$C_\Sigma(u_1, \ldots, u_n) = \Phi_\Sigma \left( \Phi^{-1}(u_1), \ldots, \Phi^{-1}(u_n) \right), \tag{1}$$

where $\Phi^{-1}$ is the inverse standard normal distribution, and $\Phi_\Sigma$ is a zero mean normal distribution with correlation matrix $\Sigma$. ∎

**Example 2.3:** The standard Gaussian distribution is mathematically convenient but limited due to its unimodal form and tail behavior. However, the Gaussian copula can give rise to complex varied distribution and offers great flexibility. As an example, Figure 1 shows two bivariate distributions that are constructed using the Gaussian copula and two different sets of univariate marginals. Generally, *any* univariate marginal, both parametric and nonparametric can be used.

Let $\varphi_\Sigma(\mathbf{x})$ denote the multivariate normal density with mean zero and covariance $\Sigma$, and let $\varphi(x)$ denote the univariate standard normal density. Using the derivative chain rule and the derivative

inverse function theorem, the Gaussian copula *density* $c(u_1, \ldots, u_n) = \frac{\partial^n C_\Sigma(u_1, \ldots, u_n)}{\partial U_1, \ldots \partial U_n}$ is

$$c(u_1, \ldots, u_n) = \varphi_\Sigma \left( \Phi^{-1}(u_1), \ldots, \Phi^{-1}(u_n) \right) \prod_i \frac{\partial \Phi^{-1}(u_i)}{\partial U_i} = \frac{\varphi_\Sigma \left( \Phi^{-1}(u_1), \ldots, \Phi^{-1}(u_n) \right)}{\prod_i \varphi(\Phi^{-1}(u_i))}.$$

For a distribution defined by a Gaussian copula $F_{\mathcal{X}}(x_1, \ldots, x_n) = C_\Sigma(F_1(x_1), \ldots, F_n(x_n))$, using $\partial U_i / \partial X_i = f_i$, we have

$$f_{\mathcal{X}}(x_1, \ldots, x_n) = \frac{\partial^n C_\Sigma(F_1(x_1), \ldots, F_n(x_n))}{\partial X_1, \ldots, \partial X_n} = \frac{\varphi_\Sigma(\tilde{x}_1, \ldots, \tilde{x}_n)}{\prod_i \varphi(\tilde{x}_i)} \prod_i f_i(x_i), \qquad (2)$$

where $\tilde{x}_i \equiv \Phi^{-1}(u_i) \equiv \Phi^{-1}(F_i(x_i))$. We will use this compact notation in the rest of the paper.

## 2.2 Copula Bayesian Networks

Let $\mathcal{G}$ be a directed acyclic graph (DAG) whose nodes correspond to the random variables $\mathcal{X} = \{X_1, \ldots, X_n\}$, and let $\mathbf{Pa}_i = \{\mathbf{Pa}_{i1}, \ldots, \mathbf{Pa}_{ik_i}\}$ be the parents of $X_i$ in $\mathcal{G}$. As for standard BNs, $\mathcal{G}$ encodes the independence statements $I(\mathcal{G}) = \{(X_i \perp NonDescendants_i \mid \mathbf{Pa}_i)\}$, where $\perp$ denotes the independence relationship, and *NonDescendants*$_i$ are nodes that are not descendants of $X_i$ in $\mathcal{G}$.

**Definition 2.4:** A Copula Bayesian Network (CBN) is a triplet $\mathcal{C} = (\mathcal{G}, \Theta_C, \Theta_f)$ that defines $f_{\mathcal{X}}(\mathbf{x})$. $\mathcal{G}$ encodes the independencies assumed to hold in $f_{\mathcal{X}}(\mathbf{x})$. $\Theta_C$ is a set of local copula functions $C_i(u_i, u_{\mathbf{pa}_{i1}}, \ldots, u_{\mathbf{pa}_{ik_i}})$ that are associated with the nodes of $\mathcal{G}$ that have at least one parent. In addition, $\Theta_f$ is the set of parameters representing the marginal densities $f_i(x_i)$ (and distributions $u_i \equiv F_i(x_i)$). The joint density $f_{\mathcal{X}}(\mathbf{x})$ then takes the form

$$f_{\mathcal{X}}(\mathbf{x}) = \prod_{i=1}^n \frac{c_i(u_i, u_{\mathbf{pa}_{i1}}, \ldots, u_{\mathbf{pa}_{ik_i}})}{\frac{\partial^K C_i(1, u_{\mathbf{pa}_{i1}}, \ldots, u_{\mathbf{pa}_{ik_i}})}{\partial U_{\mathbf{pa}_{i1}} \ldots \partial U_{\mathbf{pa}_{ik_i}}}} f_i(x_i) \equiv \prod_{i=1}^n R_{c_i}(u_i, u_{\mathbf{pa}_{i1}}, \ldots, u_{\mathbf{pa}_{ik_i}}) f_i(x_i) \qquad (3)$$

When $X_i$ has no parents in $\mathcal{G}$, $R_{c_i}(\cdot) \equiv 1$. ∎

Note that $R_{c_i}(\cdot) f_i(x_i)$ is always a valid conditional density $f(x_i \mid \mathbf{pa}_i)$, and can be easily computed. In particular, when the copula density $c(\cdot)$ in the numerator has an explicit form, so does $R_{c_i}(\cdot)$.

Elidan [2010] showed that a CBN defines a valid joint density. When the model is tree-structured, $\prod_i R_{c_i}(u_i, u_{\mathbf{pa}_{i1}}, \ldots, u_{\mathbf{pa}_{ik_i}})$ defines a valid copula so that the univariate marginals of the constructed density are $f_i(x_i)$. More generally, the marginals may be skewed. though in practice only slightly so. In this case the CBN model can be viewed as striking a balance between the *fixed marginals* and the unconstrained maximum likelihood objectives. Practically, the model leads to substantial generalization advantages (see Elidan [2010] for more details).

# 3 Nonparanormal Belief Propagation

As exemplified in Figure 1, the Gaussian copula can give rise to complex multi-modal joint distributions. When local Gaussian copulas are combined in a high-dimensional Gaussian Copula BN (GCBN), expressiveness is even greater. Yet, as we show in this section, tractable inference in this highly non-Gaussian model is possible, regardless of the form of the univariate marginals.

## 3.1 Inference for a Single Gaussian Copula

We start by showing how inference can be carried out in closed form for a single Gaussian copula. While all that is involved is a simple change of variables, the details are instructive. Let $f_{\mathcal{X}}(x_1, \ldots, x_n)$ be a density parameterized by a Gaussian copula. We start with the task of computing the multivariate marginal over a subset of variables $\mathbf{Y} \subset \mathbf{X}$. For convenience and without loss

of generality, we assume that $\mathbf{Y} = \{X_1, \ldots, X_k\}$ with $k < n$. From Eq. (2), we have

$$f_{X_1,\ldots,X_K}(x_1,\ldots,x_k) = \int_{\mathcal{R}^{n-k}} f_{\mathcal{X}}(x_1,\ldots,x_n)dx_{k+1}\ldots dx_n$$

$$= \prod_{i=1}^{k} \frac{f_i(x_i)}{\varphi(\tilde{x}_i)} \int \left[ \varphi_{\Sigma}\left(\Phi^{-1}(F_1(x_1)),\ldots,\Phi^{-1}(F_n(x_n))\right) \prod_{i=k+1}^{n} \frac{f_i(x_i)}{\varphi(\Phi^{-1}(F_i(x_i)))} \right] dx_{k+1}\ldots dx_n.$$

Changing the integral variables to $U_i$ and using $f_i = \frac{\partial U_i}{\partial X_i}$ so that $f_i(x_i)dx_i = du_i$, we have

$$f_{X_1,\ldots,X_K}(x_1,\ldots,x_k) = \prod_{i=1}^{k} \frac{f_i(x_i)}{\varphi(\tilde{x}_i)} \int_{[0,1]^{n-k}} \left[ \frac{\varphi_{\Sigma}\left(\Phi^{-1}(u_1),\ldots,\Phi^{-1}(u_n)\right)}{\prod_{i=k+1}^{n} \varphi(\Phi^{-1}(u_i))} \right] du_{k+1}\ldots du_n.$$

Changing variables once again to $\tilde{x}_i = \Phi^{-1}(u_i)$, and using $\partial \tilde{X}_i/\partial U_i = \varphi(\tilde{x}_i)^{-1}$, we have

$$f_{X_1,\ldots,X_K}(x_1,\ldots,x_k) = \prod_{i=1}^{k} \frac{f_i(x_i)}{\varphi(\tilde{x}_i)} \int_{\mathcal{R}^{n-k}} \varphi_{\Sigma}(\tilde{x}_1,\ldots,\tilde{x}_n) \, d\tilde{x}_{k+1}\ldots d\tilde{x}_n.$$

The integral on the right hand side is now a standard marginalization of a multivariate Gaussian (over $\tilde{x}_i$'s) and can be carried out in closed form.

Computation of densities conditioned on evidence $\mathbf{Z} = \mathbf{z}$ can also be easily carried out. Letting $\mathbf{W} = \mathcal{X} \setminus \{\mathbf{Z} \cup \mathbf{Y}\}$ denote non query or evidence variables, and plugging in the above, we have:

$$f_{\mathbf{Y}|\mathbf{Z}}(\mathbf{y} \mid \mathbf{z}) = \frac{\int f(\mathbf{x})d\mathbf{w}}{\iint f(\mathbf{x})d\mathbf{w}d\mathbf{y}} = \prod_{i\in\mathbf{Y}} \frac{f_i(x_i)}{\varphi(\tilde{x}_i)} \frac{\int \varphi_{\Sigma}(\tilde{x}_1,\ldots,\tilde{x}_n)\,d\tilde{\mathbf{w}}}{\iint \varphi_{\Sigma}(\tilde{x}_1,\ldots,\tilde{x}_n)\,d\tilde{\mathbf{w}}d\tilde{\mathbf{y}}}.$$

The conditional density is now easy to compute since a ratio of normal distributions is also normal. The final answer, of course, does involve $f_i(x_i)$. This is not only unavoidable but in fact desirable since we would like to retain the complexity of the desired posterior.

## 3.2 Tractability of Inference in a Gaussian CBNs

We are now ready to consider inference in a Gaussian CBN (GCBN). In this case, the joint density of Eq. (3) takes, after cancellation of terms, the following form:

$$f_{\mathcal{X}}(x_1,\ldots,x_n) = \prod_i \frac{f_i(x_i)}{\varphi(\tilde{x}_i)} \prod_i \frac{\varphi_{\Sigma_i}(\tilde{x}_i, \tilde{x}_{\mathbf{pa}_{i1}},\ldots,\tilde{x}_{\mathbf{pa}_{ik_i}})}{\varphi_{\Sigma_i^-}(\tilde{x}_{\mathbf{pa}_{i1}},\ldots,\tilde{x}_{\mathbf{pa}_{ik_i}})},$$

where $\Sigma_i^-$ is used to denote the $i$'th local covariance matrix excluding the $i$'th row and column. When $X_i$ has no parents, the ratio reduces to $\varphi(\tilde{x}_i)$. When the graph is tree structured, this density is also a copula and its marginals are $f_i(x_i)$. In this case, the same change of variables as above results in

$$f_{\widetilde{\mathcal{X}}}(\tilde{x}_1,\ldots,\tilde{x}_n) = \prod_i \frac{\varphi_{\Sigma_i}(\tilde{x}_i, \tilde{x}_{\mathbf{pa}_{i1}},\ldots,\tilde{x}_{\mathbf{pa}_{ik_i}})}{\varphi_{\Sigma_i^-}(\tilde{x}_{\mathbf{pa}_{i1}},\ldots,\tilde{x}_{\mathbf{pa}_{ik_i}})}.$$

Since a ratio of Gaussians is also a Gaussian, the entire density is Gaussian in $\tilde{x}_i$ space, and computation of any marginal $f_{\tilde{\mathbf{Y}}}(\tilde{\mathbf{y}})$ is easy. The required marginal in $x_i$ space is then recovered using

$$f_{\mathbf{Y}}(\mathbf{y}) = f_{\tilde{\mathbf{Y}}}(\tilde{\mathbf{y}}) \prod_{i\in\mathbf{Y}} \frac{f_i(x_i)}{\varphi(\tilde{x}_i)} \tag{4}$$

which essentially summarizes the detailed derivation of the previous section.

When we consider a non-tree structured CBN model, as noted in Section 2.2, the marginals may not equal $f_i(x_i)$, and the above simplification is not applicable. However, for the Gaussian case, it is always possible to estimate the local copulas in a topological order so that the univariate marginals are equal to $f_i(x_i)$ (the model in this case is equivalent to the distribution-free continuous Bayesian belief net model [Kurowicka and Cooke, 2005]). It follows that, for any structure,

**Corollary 3.1:** *The complexity of inference in a Gaussian CBN model is the same as that of inference in a multivariate Gaussian model of the same structure.*

**Algorithm 1:** Nonparanormal Belief Propagation (NPNBP) for general CBNs.

**Input**: $\{f_k(x_k)\}$ for all $i$, $\Sigma_i$ for all nodes with parents. **Output**: belief $b_S(\mathbf{x}_S)$ for each cluster $S$.

---

$\mathcal{CG} \leftarrow$ a valid cluster graph over the following potentials for all nodes $i$ in the graph

- $\varphi_{\Sigma_i}(\tilde{x}_i, \tilde{x}_{\mathbf{pa}_{i1}}, \ldots, \tilde{x}_{\mathbf{pa}_{ik_i}})$
- $1/\varphi_{\Sigma_i^-}(\tilde{x}_{\mathbf{pa}_{i1}}, \ldots, \tilde{x}_{\mathbf{pa}_{ik_i}})$

**foreach** *cluster S in* $\mathcal{CG}$                               `// use black-box GaBP in `$\tilde{x}_i$` space`

    |  $b_G(\tilde{\mathbf{x}}_S) \leftarrow$ GaBP belief over cluster $S$.

**foreach** *cluster S in* $\mathcal{CG}$                                   `// change to `$x_i$` space`

    |  $b_S(\mathbf{x}_S) = b_G(\tilde{\mathbf{x}}_S) \prod_{i \in S} \frac{f_i(x_i)}{\varphi(\tilde{x}_i)}$

---

While mathematically this conclusion is quite straightforward, the implications are significant. A GCBN model is the *only* general purpose non-Gaussian continuous graphical model for which exact inference is tractable. At the same time, as is demonstrated in our experimental evaluation, the model is able to capture complex distributions well both qualitatively and quantitatively.

A final note is worthwhile regarding the (possibly conditional) marginal density. As can be expected, the result of Eq. (4) includes $f_i(x_i)$ terms for all variables that have not been marginalized out. As noted, this is indeed desirable as we would like to preserve the complexity of the density in the marginal computation. The marginal term, however, is now in low dimension so that quantities of interest (e.g., expectation) can be readily computed using naive grid-based evaluation or, if needed, using more sophisticated sampling schemes (see, for example, [Robert and Cassella, 2005]).

### 3.3 Belief Propagation for Gaussian CBNs

Given the above observations, performing inference in a Gaussian CBN (GCBN) appears to be a solved problem. However, inference in large-scale models can be problematic even in the Gaussian case. First, the large joint covariance matrix may be ill conditioned and inverting it may not be possible. Second, matrix inversion can be slow when dealing with domains of sufficient dimension.

A possible alternative is to consider the popular belief propagation algorithm [Pearl, 1988]. For a tree structured model represented as a product of singleton $\Psi_i$ and pairwise $\Psi_{ij}$ factors, the method relies on the recursive computation of "messages"

$$m_{i \to j}(x_j) \leftarrow \alpha \int [\Psi_{ij}(x_i, x_j)\Psi_i(x_i) \prod_{k \in N(i) \setminus j} m_{k \to i}(x_i)] dx_i,$$

where $\alpha$ is a normalization factor and $N(i)$ are the indices of the neighbor nodes of $X_i$.

In the case of a GCBN model, performing belief propagation may seem difficult since $\Psi_i(x_i) \equiv f_i(x_i)$ can have a complex form. However, the change of variables used in the previous section applies here as well. That is, one can perform inference in $\tilde{x}_i$ space using standard Gaussian BP (GaBP) [Weiss and Freeman, 2001], and then perform the needed change of variables. In fact, this is true regardless of the structure of the graph so that *loopy* GaBP can also be used to perform *approximate* computations for a general GCBN model in $\tilde{x}_i$ space. The approach is summarized in Algorithm 1, where we assume access to a black-box GaBP procedure and a cluster graph construction algorithm. In our experiments we simply use a Bethe approximation construction (see [Koller and Friedman, 2009] for details on BP, GaBP and the cluster graph construction).

Generally, little can be said about the convergence of loopy BP or its variants, particularly for non-Gaussian domains. Appealingly, the form of our NPNBP algorithm implies that its convergence can be phrased in terms of standard Gaussian BP convergence. In particular:

- **Observation 1**: NPNBP converges whenever GaBP converges for the model defined by $\prod_i R_{c_i}$.
- **Observation 2**: Convergence of NPNBP depends only on the covariance matrices $\Sigma_i$ that parameterize the local copula and *does not* depend on the univariate form.

It follows that convergence conditions identified for GaBP [Rusmevichientong and Roy, 2000, Weiss and Freeman, 2001, Malioutov et al., 2006] carry over to NPNBP for CBN models.

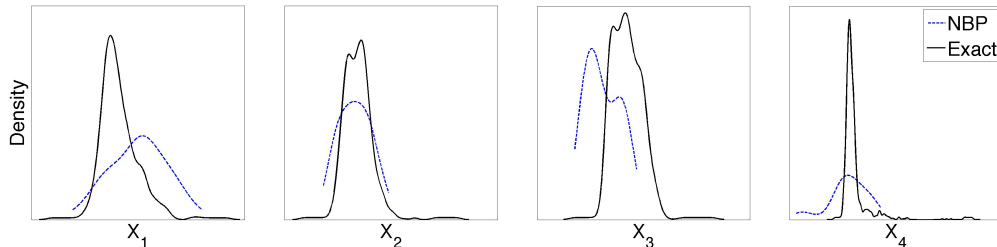

Figure 2: Exact vs. Nonparametric BP marginals for the GCBN model learned from the wine quality dataset. Shown are the marginal densities for the first four variables.

## 4 Experimental Evaluation

We now consider the merit of using our NPNBP algorithm for performing inference in a a Gaussian CBN (GCBN) model. We learned a tree structured GCBN using a standard Chow-Liu approach [Chow and Liu, 1968], and a model with up to two parents for each variable using standard greedy structure search. In both cases we use the Bayesian Information Criterion (BIC) [Schwarz, 1978] to guide the structure learning algorithm. For the univariate densities, we use a standard Gaussian kernel density estimator (see, for example, [Bowman and Azzalini, 1997]). Using an identical procedure, we learn a linear Gaussian BN baseline where $X_i \sim N(\alpha \mathbf{pa}_i, \sigma_i)$ so that each variable $X_i$ is normally distributed around a linear combination of its parents $\mathbf{Pa}_i$ (see [Koller and Friedman, 2009] for details on this standard approach to structure learning).

For the GCBN model, we also compare to Nonparametric BP (NBP) [Sudderth et al., 2010a] using D. Bickson's code [Bickson, 2008] and A. Ihlers KDE Matlab package (*http://www.ics.uci.edu/ ihler/code/kde.html*), which relies on a mixture of Gaussians for message representation. In this case, since our univariate densities are constructed using Gaussian kernels, there is no approximation in the NBP representation and all approximations are due to message computations. To carry out message products, we tried all 7 sampling-based methods available in the KDE package. In the experiments below we use only the multiresolution sequential Gibbs sampling method since all other approaches resulted in numerical overflows even for small domains.

### 4.1 Qualitative Assessment

We start with a small domain where the qualitative nature of the inferred marginals is easily explored, and consider performance and running time in more substantial domains in the next section. We use the wine quality data set from the UCI repository which includes 1599 measurements of 11 physiochemical properties and a quality variable of red "Vinho Verde" [Cortez et al., 2009].

We first examine a tree structured GCBN model where our NPNBP method allows us to perform exact marginal computations. Figure 2 compares the first four marginals to the ones computed by the NBP method. As can be clearly seen, although the NBP marginals are not nonsensical, they are far from accurate (results for the other marginals in the domain are similar). Quantitatively, each NBP marginal is 0.5 to 1.5 bits/instance less accurate than the exact ones. Thus, the accuracy of NPNBP in this case is approximately twice that of NBP *per variable*, amounting to a substantial *per sample* advantage. We also note that NBP was approximately an order of magnitude slower than NPNBP in this domain. In the larger domains considered in the next section, NBP proved impractical.

Figure 3 demonstrates the quality of the *bivariate* marginals inferred by our NPNBP method relative to the ones of a linear Gaussian BN model where inference can also be carried out efficiently. The middle panel shows a Gaussian distribution constructed only over the two variables and is thus an upper bound on the quality that we can expect from a linear Gaussian BN. Clearly, the Gaussian representation is not sufficiently flexible to reasonably capture the distribution of the true samples (left panel). In contrast, the bivariate marginals computed by our algorithm (right panel) demonstrate the power of working with a copula-based construction and an effective inference procedure: in both cases the inferred marginals capture the non-Gaussian distributions quite accurately. Results were qualitatively similar for all other variable pairs (except for the *few* cases that are approximately Gaussian in the original feature space and for which all models are equally beneficial).

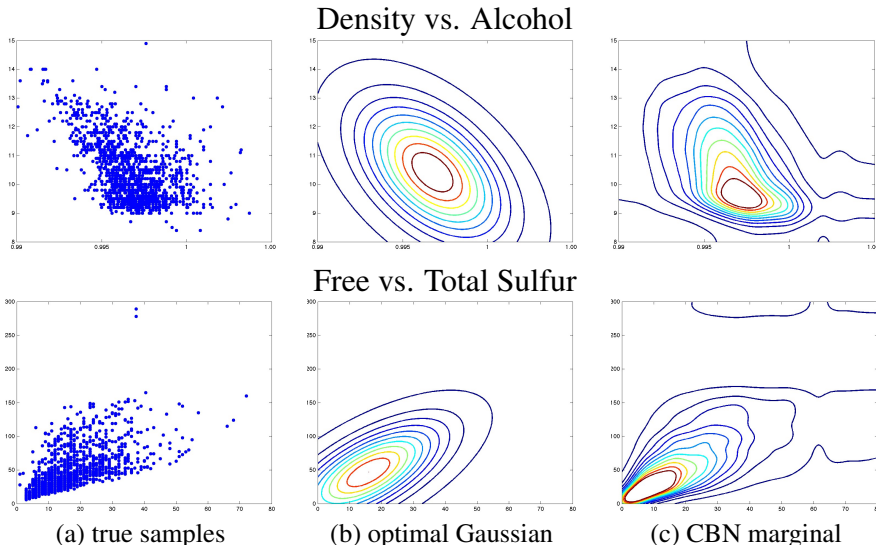

(a) true samples     (b) optimal Gaussian     (c) CBN marginal

Figure 3: The bivariate density for two pairs of variables in a tree structured GCBN model learned from the wine quality dataset. (a) empirical samples; (b) maximum likelihood Gaussian density; (c) exact GCBN marginal computed using our NPNBP algorithm.

In Figure 4 we repeat the comparison for another pair of variables in a non-tree GCBN (as before, results were qualitatively similar for all pairs of variables). In this setting, the bivariate marginal computed by our algorithm (d) is approximate and we also compare to the exact marginal (c). As in the case of the tree-structured model, the GCBN model captures the true density quite accurately, even for this multi-modal example. NPNBP dampens some of this accuracy and results in marginal densities that have the correct overall structure but with a reduced variance. This is not surprising since it is well known that GaBP leads to reduced variances [Weiss and Freeman, 2001]. Nevertheless, the *approximate* result of NPNBP is clearly better than the *exact* Gaussian model, which assigns very low probability to regions of high density (along the main vertical axis of the density).

Finally, Figure 5(left) shows the NPNBP vs. the exact expectations. As can be seen, the inferred values are quite accurate and it is plausible that the differences are due to numerical round-offs. Thus, it is possible that, similarly to the case of standard GaBP [Weiss and Freeman, 2001], the inferred expectations are *theoretically* exact. The proof for the GaBP case, however, does not carry over to the CBN setting and shedding theoretical light on this issue remains a future challenge.

### 4.2 Quantitative Assessment

We now consider several substantially larger domains with 100 to almost 2000 variables. For each domain we learn a tree structured GCBN, and justify the need for the expressive copula-based model by reporting its average generalization advantage in terms of log-loss/instance over a standard linear Gaussian model. We justify the use of NPNBP for performing inference by comparing the running time of NPNBP to exact computations carried out using matrix inversion. For all datasets, we performed 10-fold cross-validation and report average results. We use the following datasets:

- **Crime** (UCI repository). 100 variables relating to crime ranging from household size to fraction of children born outside of a marriage, for 1994 communities across the U.S.
- **SP500**. Daily changes of value of the 500 stocks comprising the Standard and Poor's index (S&P 500) over a period of one year.
- **Gene**. A compendium of gene expression experiments used in [Marion et al., 2004]. We chose genes that have only 1, 2, and 3 missing values and only use full observations. This resulted in datasets with 765, 1400, and 1945 variables (genes), and 1088, 956, and 876 samples, respectively.

For the 100 variable **Crime** domain, average test advantage of the GCBN model over the linear Gaussian one was 0.39 bits/instance *per variable* (as in [Elidan, 2010]). For the 765 variable **Gene** expression domain the advantage was around 0.1 bits/instance/variable (results were similar for the

Sugar level vs. Density

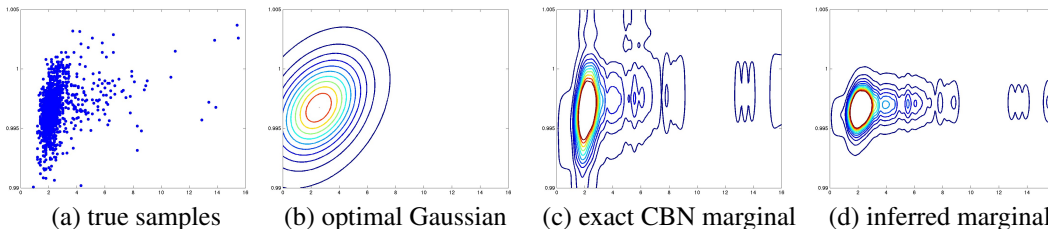

(a) true samples      (b) optimal Gaussian      (c) exact CBN marginal      (d) inferred marginal

Figure 4: The bivariate density for a pair of variables in a non-tree GCBN model learned from the wine quality dataset. (a) empirical samples; (b) maximum likelihood Gaussian density; (c) exact CBN marginal; (d) marginal density computed by our NPNBP algorithm.

Figure 5: (left) exact vs. NPNBP expected values. (right) speedup relative to matrix inversion for a tree structured GCBN model. 765,1400,1945 correspond to the three different datasets extracted from the gene expression compendium.

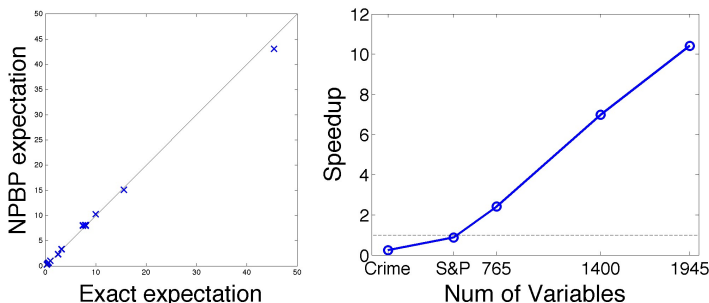

other gene expression datasets). In both cases, the differences are dramatic and each instance is many orders of magnitude more likely given a GCBN model. For the **SP500** domain, evaluation of the linear Gaussian model resulted in numerical overflows (due to the scarcity of the training data), and the advantage of he GCBN cannot be quantified. These generalization advantages make it obvious that we would like to perform efficient inference in a GCBN model.

As discussed, a GCBN model is itself tractable in that inference can be carried out by first constructing the inverse covariance matrix over all variables and then inverting it so as to facilitate marginalization. Thus, using our NPNBP algorithm can only be justified practically. Figure 5(right) shows the speedup of NPNBP relative to inference based on matrix inversion for the different domains. Although NPNBP is somewhat slower for the small domains (in which inference is carried out in less than a second), the speedup of NPNBP reaches an order of magnitude for the larger gene expression domain. Appealingly, the advantage of NPNBP grows with the domain size due to the growth in complexity of matrix inversion. Finally, we note that we used a Matlab implementation where matrix inversion is highly optimized so that the gains reported are quite conservative.

## 5    Summary

We presented Nonparanormal Belief Propagation (NPNBP), a propagation-based algorithm for performing highly efficient inference in a powerful class of graphical models that are based on the Gaussian copula. To our knowledge, ours is the first inference method for an expressive continuous non-Gaussian representation that, like ordinary GaBP, is both highly efficient and provably correct for tree structured models. Appealingly, the efficiency and convergence properties of our method do not depend on the choice of univariate marginals, even when a nonparametric representation is used.

The Gaussian copula is a powerful model widely used to capture complex phenomenon in fields ranging from mainstream economics (e.g., Embrechts et al. [2003]) to flood analysis [Zhang and Singh, 2007]. Recent probabilistic graphical models that build on the Gaussian copula open the door for new high-dimensional non-Gaussian applications [Kirshner, 2007, Liu et al., 2010, Elidan, 2010, Wilson and Ghahramani, 2010]. Our method offers the inference tools to make this practical.

## Acknowledgements

G. Elidan and C. Cario were supported in part by an ISF center of research grant. G. Elidan was also supported by a Google grant. Many thanks to O. Meshi and A. Globerson for their comments on an earlier draft.

## References

D. Bickson. *Gaussian Belief Propagation: Theory and Application*. PhD thesis, The Hebrew University of Jerusalem, Jerusalem, Israel, 2008.

A. Bowman and A. Azzalini. *Applied Smoothing Techniques for Data Analysis*. Oxford Press, 1997.

C. K. Chow and C. N. Liu. Approximating discrete probability distributions with dependence trees. *IEEE Trans. on Info. Theory*, 14:462–467, 1968.

P. Cortez, A. Cerdeira, F. Almeida, T. Matos, and J. Reis. Modeling wine preferences by data mining from physicochemical properties. *Decision Support Systems*, 47(4):547–553, 2009.

G. Elidan. Copula bayesian networks. In *Neural Information Processing Systems (NIPS)*, 2010.

P. Embrechts, F. Lindskog, and A. McNeil. Modeling dependence with copulas and applications to risk management. *Handbook of Heavy Tailed Distributions in Finance*, 2003.

A. Globerson and T. Jaakkola. Fixing max-product: Convergent message passing algorithms for map lp-relaxations. In *Neural Information Processing Systems (NIPS)*, 2007.

T. Heskes. On the uniqueness of loopy belief propagation fixed points. *Neural Comp*, 16:2379–2413, 2004.

A. Ihler and D. McAllester. Particle belief propagation. In *Conf on AI and Statistics (AISTATS)*, 2009.

S. Kirshner. Learning with tree-averaged densities and distributions. In *Neural Info Proc Systems (NIPS)*, 2007.

D. Koller and N. Friedman. *Probabilistic Graphical Models: Principles and Techniques*. The MIT Press, 2009.

D. Kurowicka and R. M. Cooke. Distribution-free continuous bayesian belief nets. In *Selected papers based on the presentation at the international conference on mathematical methods in reliability (MMR)*, 2005.

H. Liu, J. Lafferty, and L. Wasserman. The nonparanormal: Semiparametric estimation of high dimensional undirected graphs. *Journal of Machine Learning Research*, 2010.

D. Malioutov, J. Johnson, and A. Willsky. Walk-sums and belief propagation in gaussian graphical models. *Journal of Machine Learning Research*, 7:2031–2064, 2006.

R. Marion, A. Regev, E. Segal, Y. Barash, D. Koller, N. Friedman, and E. O'Shea. Sfp1 is a stress- and nutrient-sensitive regulator of ribosomal protein gene expression. *Proc Natl Acad Sci U S A*, 101(40):14315–22, 2004.

R. McEliece, D. McKay, and J. Cheng. Turbo decoding as an instance of pearl's belief propagation algorithm. *IEEE Journal on Selected Areas in Communication*, 16:140–152, 1998.

T. P. Minka. Expectation propagation for approximate Bayesian inference. In *Proc. Conference on Uncertainty in Artificial Intelligence (UAI)*, pages 362–369, 2001.

J. Mooij and B. Kappen. Sufficient conditions for convergence of loopy belief propagation. In *Proc. Conference on Uncertainty in Artificial Intelligence (UAI)*, 2005.

K. Murphy and Y. Weiss. Loopy belief propagation for approximate inference: An empirical study. In *Proc. Conference on Uncertainty in Artificial Intelligence (UAI)*, pages 467–475, 1999.

R. Nelsen. *An Introduction to Copulas*. Springer, 2007.

J. Pearl. *Probabilistic Reasoning in Intelligent Systems*. Morgan Kaufmann, 1988.

C. P. Robert and G. Casella. *Monte Carlo Statistical Methods (Springer Texts in Statistics*. Springer, 2005.

P. Rusmevichientong and B. Van Roy. An analysis of belief propagation on the turbo decoding graph with gaussian densities. *IEEE Transactions on Information Theory*, 47:745–765, 2000.

G. Schwarz. Estimating the dimension of a model. *Annals of Statistics*, 6:461–464, 1978.

A. Sklar. Fonctions de repartition a n dimensions et leurs marges. *Publications de l'Institut de Statistique de L'Universite de Paris*, 8:229–231, 1959.

L. Song, A. Gretton, D. Bickson, Y. Low, and C. Guestrin. Kernel belief propagation. In *Conference on Artificial Intelligence and Statistics (AIStats)*, 2011.

E.B. Sudderth, A.T. Ihler, M. Isard, W.T. Freeman, and A.S. Willsky. Nonparametric belief propagation. *Communications of the ACM*, 53(10):95–103, 2010a.

Erik Sudderth, Alexander Ihler, Michael Isard, William Freeman, and Alan Willsky. Nonparametric belief propagation. *Communications of the ACM*, 53(10):95–103, October 2010b.

Y. Weiss and W. Freeman. Correctness of belief propagation in gaussian graphical models of arbitrary topology. *Neural Computation*, 13:2173–2200, 2001.

W. Wiegerinck and T. Heskes. Fractional belief propagation. In *Neural Information Processing Systems 15*, Cambridge, Mass., 2003. MIT Press.

A. Wilson and Z. Ghahramani. Copula processes. In *Neural Information Processing Systems (NIPS)*, 2010.

J. S. Yedidia, W. T. Freeman, and Y. Weiss. Generalized belief propagation. In *Neural Information Processing Systems 13*, pages 689–695, Cambridge, Mass., 2001. MIT Press.

L. Zhang and V. Singh. Trivariate flood frequency analysis using the Gumbel-Hougaard copula. *Journal of Hydrologic Engineering*, 12, 2007.

